# MODELING SMALL OSCILLATING

# BIOLOGICAL NETWORKS IN ANALOG VLSI

Sylvie Ryckebusch, James M. Bower, and Carver Mead
California Institute of Technology
Pasadena, CA 91125

## ABSTRACT

We have used analog VLSI technology to model a class of small oscillating biological neural circuits known as central pattern generators (CPG). These circuits generate rhythmic patterns of activity which drive locomotor behaviour in the animal. We have designed, fabricated, and tested a model neuron circuit which relies on many of the same mechanisms as a biological central pattern generator neuron, such as delays and internal feedback. We show that this neuron can be used to build several small circuits based on known biological CPG circuits, and that these circuits produce patterns of output which are very similar to the observed biological patterns.

To date, researchers in applied neural networks have tended to focus on mammalian systems as the primary source of potentially useful biological information. However, invertebrate systems may represent a source of ideas in many ways more appropriate, given current levels of engineering sophistication in building neural-like systems, and given the state of biological understanding of mammalian circuits. Invertebrate systems are based on orders of magnitude smaller numbers of neurons than are mammalian systems. The networks we will consider here, for example, are composed of about a dozen neurons, which is well within the demonstrated capabilities of current hardware fabrication techniques. Furthermore, since much more detailed structural information is available about these systems than for most systems in higher animals, insights can be guided by real information rather than by guesswork. Finally, even though they are constructed of small numbers of neurons, these networks have numerous interesting and potentially even useful properties.

## CENTRAL PATTERN GENERATORS

Of all the invertebrate neural networks currently being investigated by neurobiologists, the class of networks known as central pattern generators (CPGs) may be especially worthy of attention. A CPG is responsible for generating oscillatory neural activity that governs specific patterns of motor output, and can generate its pattern of activity when isolated from its normal neuronal inputs. This prop-

erty, which greatly facilitates experiments, has enabled biologists to describe several CPGs in detail at the cellular and synaptic level. These networks have been found in all animals, but have been extensively studied in invertebrates [Selverston, 1985].

We chose to model several small CPG networks using analog VLSI technology. Our model differs from most computer simulation models of biological networks [Wilson and Bower, in press] in that we did not attempt to model the details of the individual ionic currents, nor did we attempt to model each known connection in the networks. Rather, our aim was to determine the basic functionality of a set of CPG networks by modeling them as the minimum set of connections required to reproduce output qualitatively similar to that produced by the real network under certain conditions.

## MODELING CPG NEURONS

The basic building block for our model is a general purpose CPG neuron circuit. This circuit, shown in Figure 1, is our model for a typical neuron found in central pattern generators, and contains some of the essential elements of real biological neurons. Like real neurons, this model integrates current and uses positive feedback to output a train of pulses, or action potentials, whose frequency depends on the magnitude of the current input. The part of the circuit which generates these pulses is shown in Figure 2a [Mead, 1989].

The second element in the CPG neuron circuit is the synapse. In Figure 1, each pair of transistors functions as a synapse. The p-well transistors are excitatory synapses, whereas the n-well transistors are inhibitory synapses. One of the transistors in the pair sets the strength of the synapse, while the other transistor is the input of the synapse. Each CPG neuron has four different synapses.

The third element of our model CPG neuron involves temporal delays. Delays are an essential element in the function of CPGs, and biology has evolved many different mechanisms to introduce delays into neural networks. The membrane capacitance of the cell body, different rates of chemical reactions, and axonal transmission are just a few of the mechanisms which have time constants associated with them. In our model we have included synaptic delay as the principle source of delay in the network. This is modeled as an RC delay, implemented by the follower-integrator circuit shown in Figure 2b [Mead, 1989]. The time constant of the delay is a function of the conductance of the amplifier, set by the bias G. A multiple time constant delay line is formed by cascading several of these elements. Our neuron circuit uses a delay line with three time constants. The synapses which are before the delay element are *slow* synapses, whereas the undelayed synapses are *fast* synapses.

We fabricated the circuit shown in Figure 1 using CMOS, VLSI technology. Several of these circuits were put on each chip, with all of the inputs and controls going out to pads, so that these cells could be externally connected to form the network of interest.

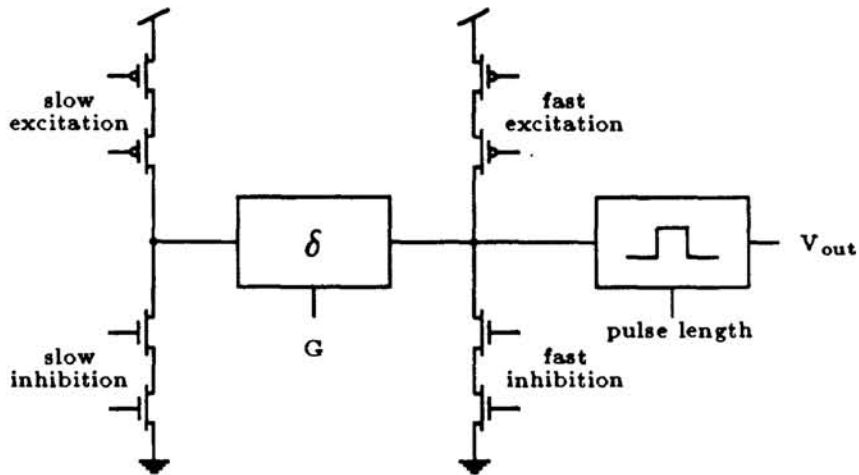

**Figure 1.** The CPG neuron circuit.

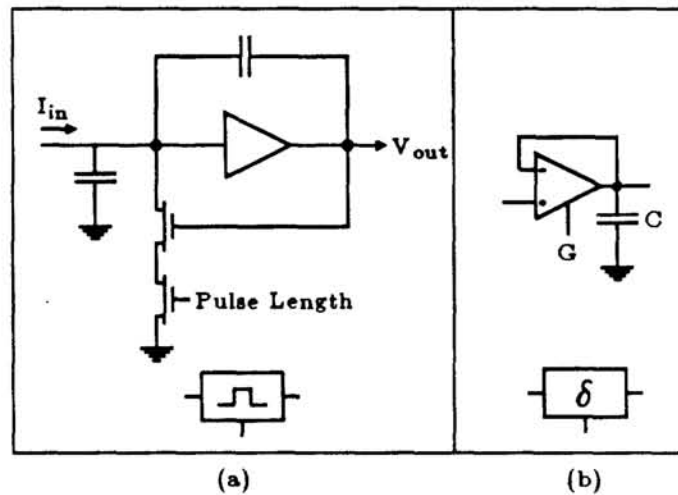

(a)                              (b)

**Figure 2.** (a). The neuron spike-generating circuit. (b). The follower-integrater circuit. Each delay box $\delta$ contains a delay line formed by three follower-integrater circuits.

## The Endogenous Bursting Neuron

One type of cell which has been found to play an important role in many oscillatory circuits is the endogenous bursting neuron. This type of cell has an intrinsic oscillatory membrane potential, enabling it to produce bursts of action potentials at rhythmic intervals. These cells have been shown to act both as external "pacemakers" which set the rhythm for the CPG, or as an integral part of a central pattern generator. Figure 3a shows the output from a biological endogenous bursting neuron. Figure 3b demonstrates how we can configure our CPG neuron to be an endogenous bursting neuron. The delay element in the cell must have three time constants in order for this circuit to oscillate stably. Note that in the circuit, the

cell has internal negative feedback. Since real neurons don't actually make synaptic connections onto themselves, this connection should be thought of as representing an internal molecular or ionic mechanism which results in feedback within the cell.

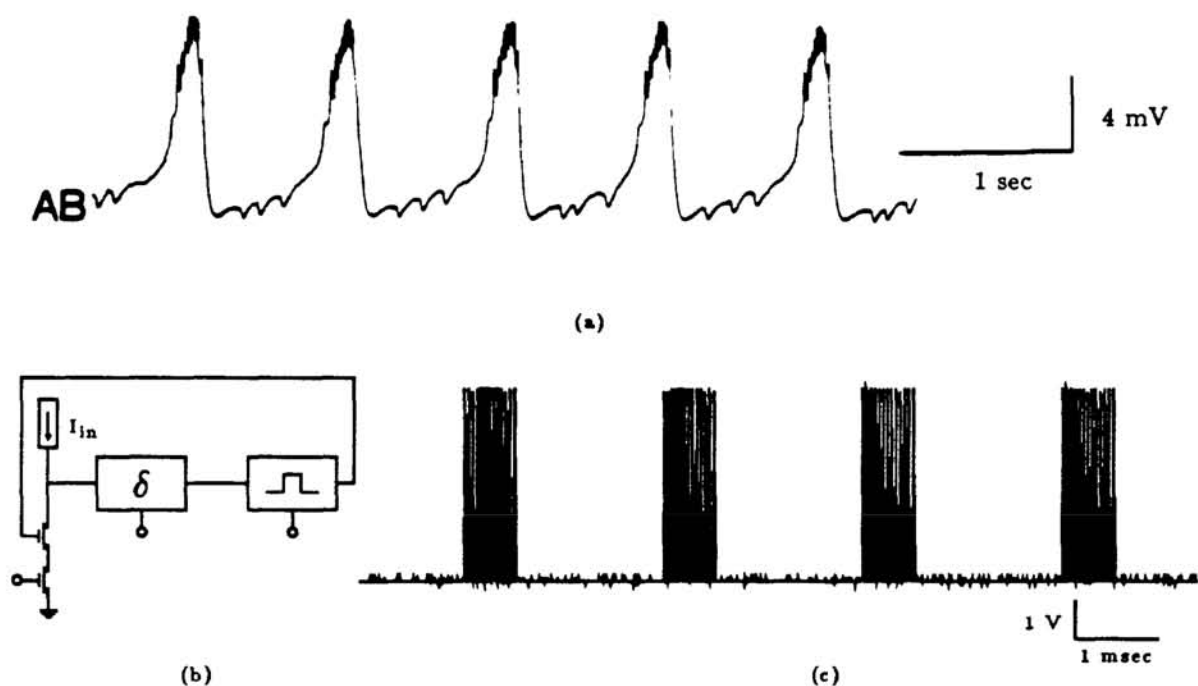

Figure 3. (a). The output from the AB cell in the lobster stomatogastric ganglion CPG [Eisen and Marder, 1982]. This cell is known to burst endogenously. (b). The CPG neuron circuit configured as an endogenous bursting neuron and (c) the output from this circuit.

## Postinhibitory Rebound

A neuron configured to be an endogenous burster also exhibits another property common to many neurons, including many CPG neurons. This property, illustrated in Figures 4a and 4b, is known as postinhibitory rebound (PIR). Neurons with this property display increased excitation for a certain period of time following the release of an inhibitory influence. This property is a useful one for central pattern generator neurons to have, because it enables patterns of oscillations to be reset following the release of inhibition.

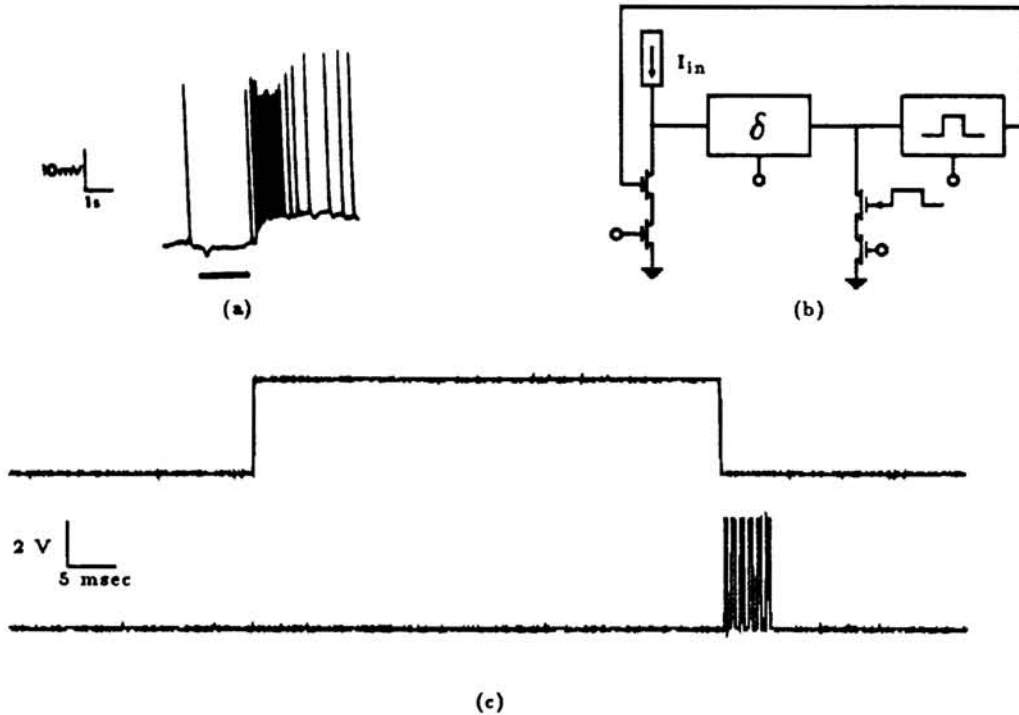

(a)

(b)

(c)

**Figure 4.** (a) The output of a ganglion cell of the mudpuppy retina exhibiting postinhibitory rebound [Miller and Dacheux, 1976]. The bar under the trace indicates the duration of the inhibition. (b) To exhibit PIR in the CPG neuron circuit, we inhibit the cell with the square pulse shown in (c). When the inhibition is released, the circuit outputs a brief burst of pulses.

## MODELING CENTRAL PATTERN GENERATORS

### The Lobster Stomatogastric Ganglion

The stomatogastric ganglion is a CPG which controls the movement of the teeth in the lobster's stomach. This network is relatively complex, and we have only modeled the relationships between two of the neurons in the CPG (the PD and LP cells) which have a kind of interaction found in many CPGs known as reciprocal inhibition (Figure 5a). In this case, each cell inhibits the other, which produces a pattern of output in which the cells fire alternatively (Figure 5b). Note that in the absence of external input, a mechanism such as postinhibitory rebound must exist in order for a cell to begin firing again once it has been released from inhibition.

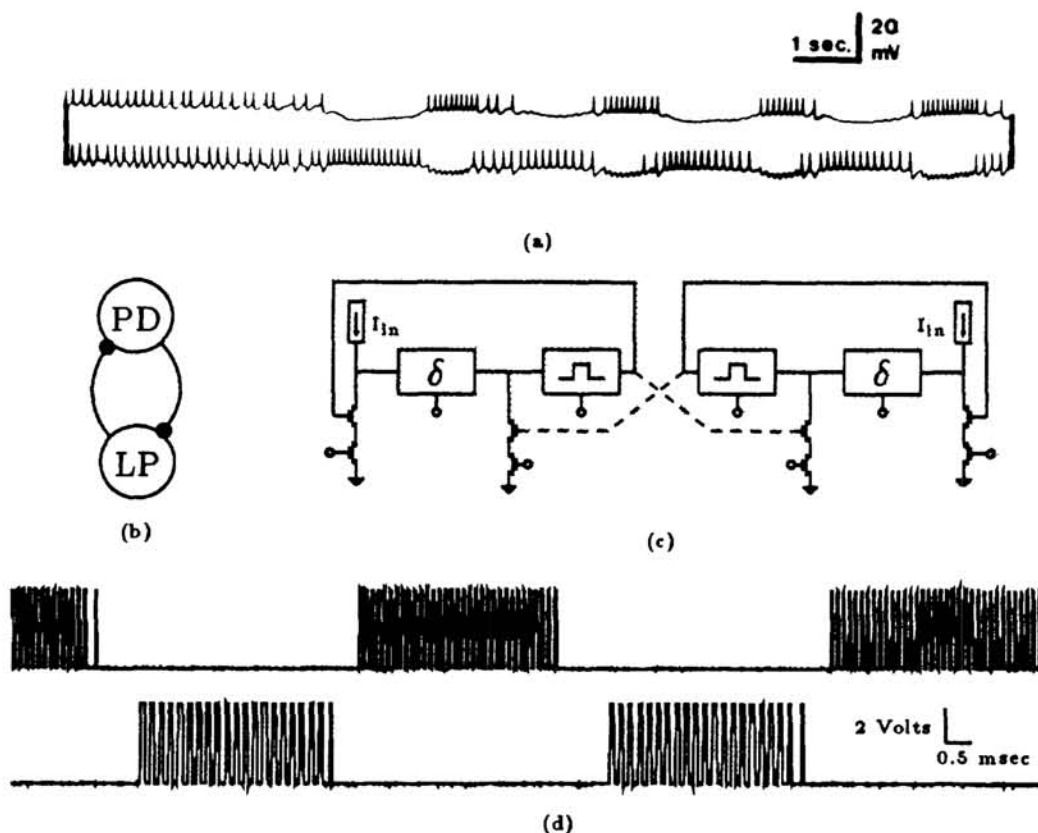

**Figure 5.** (a) Output from the PD and LP cells in the lobster stomatogastric ganglion [Miller and Selverston, 1985]. (c) and (d) demonstrate reciprocal inhibition with two CPG neuron circuits.

## The Locust Flight CPG

A CPG has been shown to play an important role in producing the motor pattern for flight in the locust [Robertson and Pearson, 1985]. Two of the cells in the CPG, the 301 and 501 cells, fire bursts of action potentials as shown in Figure 6a. The 301 cell is active when the wings of the locust are elevated, whereas the 501 cell is active when the wings are depressed. The phase relationship between the two cells is very similar to the reciprocal inhibition pattern just discussed, but the circuit that produces this pattern is quite different. The connections between these two cells are shown in Figure 6b. The 301 cell makes a delayed excitatory connection onto the 501 cell, and the 501 cell makes fast inhibitory contact with the 301 cell. Therefore, the 301 cell begins to fire, and after some delay, the 501 cell is activated. When the 501 cell begins to fire, it immediately shuts off the 301 cell. Since the 501 cell is no longer receiving excitatory input, it will eventually stop firing, releasing the 301 cell from inhibition. The cycle then repeats. This same circuit has been reproduced with our model in Figures 6c and 6d.

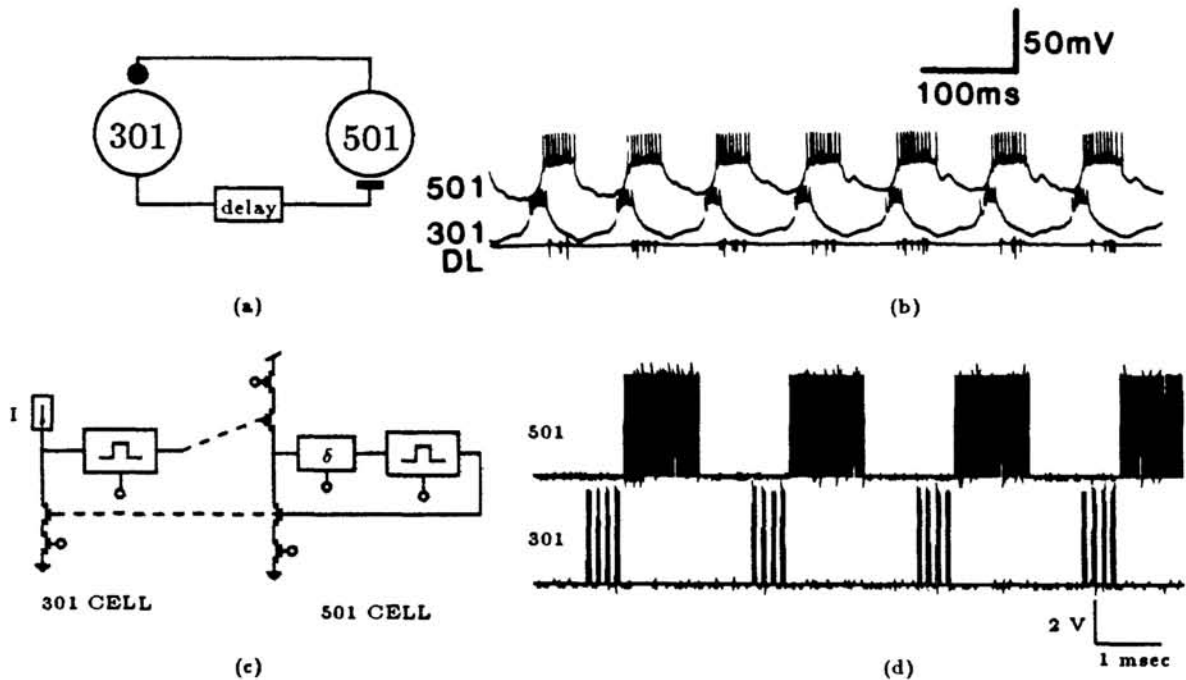

**Figure 6.** (a) The 301 and 501 cells in the locust flight CPG [Robertson and Pearson, 1985]. (b) Simultaneous intracellular recordings of 301 and 501 during flight. (c) The model circuit and (d) its output.

## The Tritonia Swim CPG

One of the best studied central pattern generators is the CPG which controls the swimming in the small marine mollusc Tritonia. This CPG was studied in great detail by Peter Getting and his colleagues at the University of Iowa, and it is one of the few biological neural networks for which most of the connections and the synaptic parameters are known in detail. Tritonia swims by making alternating dorsal and ventral flexions. The dorsal and ventral motor neurons are innervated by the DSI and VSI cells, respectively. Shown in Figure 7a and 7b is a simplified schematic diagram for the network and the corresponding output. The DSI and VSI cells fire out of phase, which is consistent with the alternating nature of the animal's swimming motion. The basic circuit consists of reciprocal inhibition between DSI and VSI paralleled by delayed excitation via the C2 cell. The DSI and VSI cells fire out of phase, and the DSI and C2 cells fire in phase. Swimming is initiated by sensory stimuli which feed into DSI and cause it to begin to fire a burst of impulses. DSI inhibits VSI, and at the same time excites C2. C2 has excitatory synapses on VSI; however, the initial response of VSI neurons is delayed. VSI then fires, during which there is inhibition by VSI of C2 and DSI. During this period, VSI no longer receives excitatory input from C2, and hence the VSI firing rate declines; DSI is therefore released from inhibition, and is ready to fire again to initiate a new cycle. Figure 7c and 8 show the model circuit which is identical to the circuit

shown in Figure 7a, and the output from this circuit. Note that although the model output closely resembles the biological data, there are small differences in the phase relationships between the cells which can be accounted for by taking into account other connections and delays in the circuit not currently incorporated in our model.

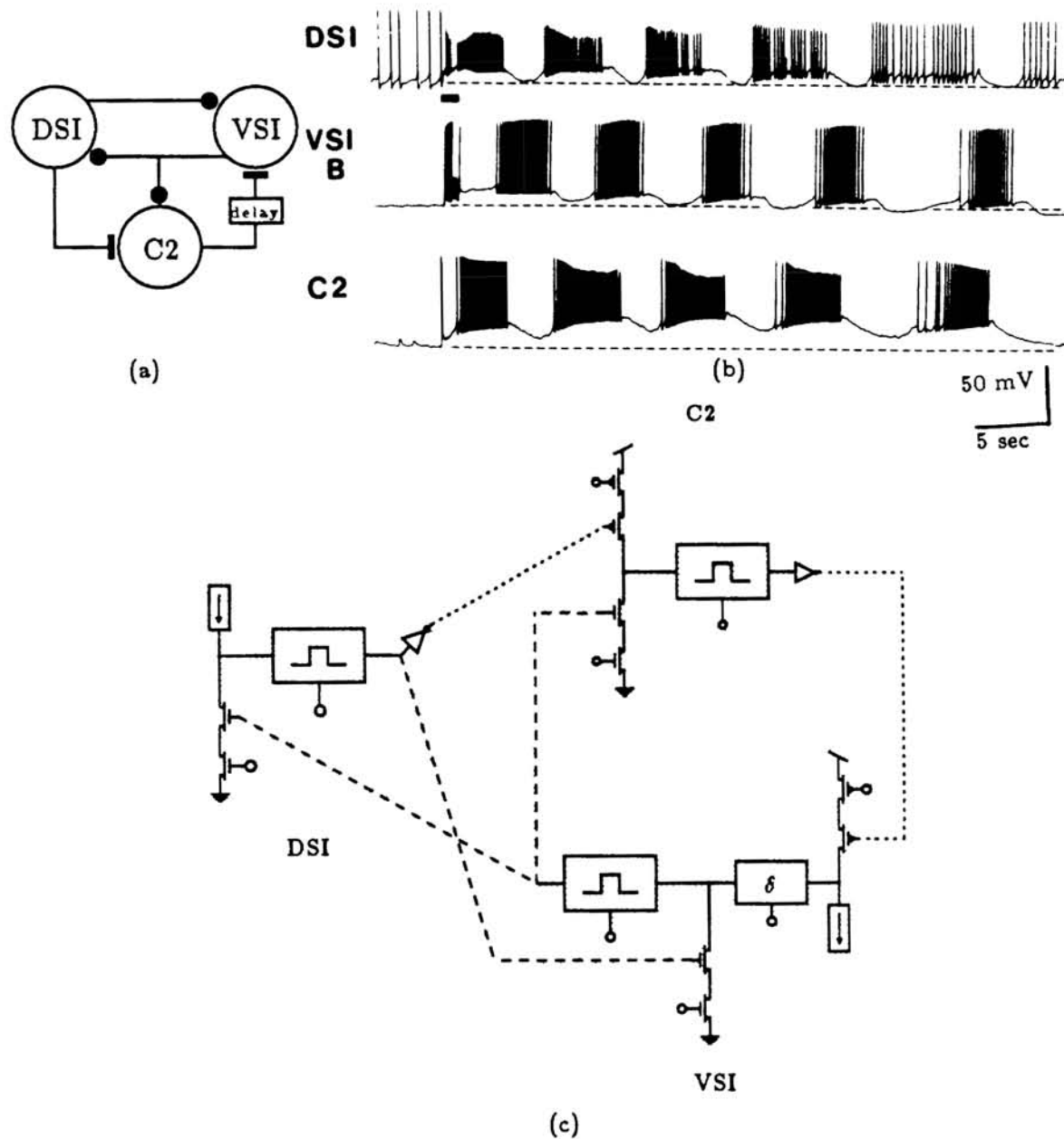

Figure 7. (a) Simplified schematic diagram of the Tritonia CPG (which actually has 14 cells) and (b) output from the three types of cells in the circuit.(c) The model circuit.

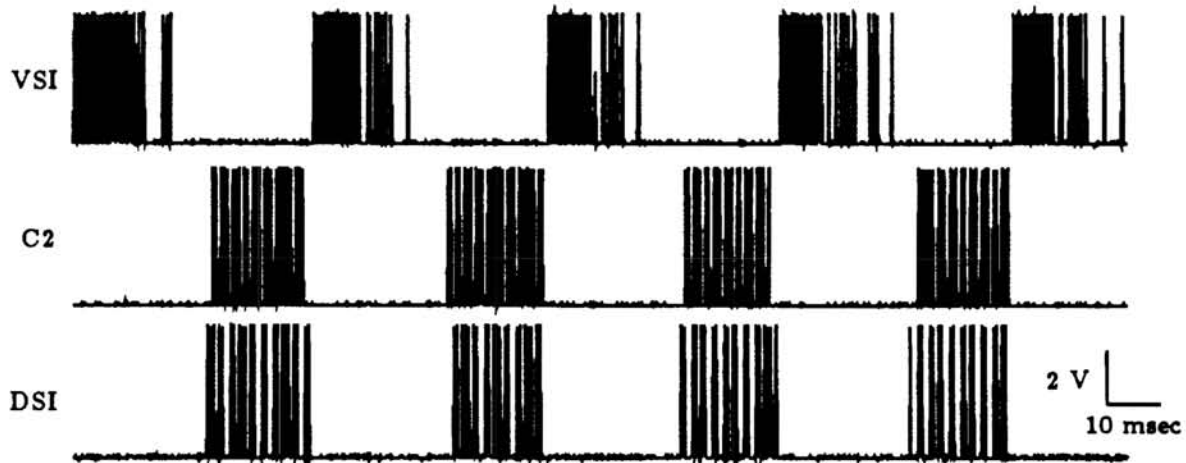

**Figure 8.** Output from the circuit shown in Figure 7c.

## CONCLUSIONS

One may ask why it is interesting to model these systems in analog VLSI, or, for that matter, why it is interesting to model invertebrate networks altogether. Analog VLSI is a very nice medium for this type of modeling, because in addition to being compact, it runs in real time, eliminating the need to wait hours to get the results of a simulation. In addition, the electronic circuits rely on the same physical principles as neural processes (including gain, delays, and feedback), allowing us to exploit the inherent properties of the medium in which we work rather than having to explicitly model them as in a digital simulation.

Like all models, we hope that this work will help us learn something about the systems we are studying. But in addition, although invertebrate neural networks are relatively simple and have small numbers of cells, the behaviours of these networks and animals can be fairly complex. At the same time, their small size allows us to understand how they are engineered in detail. Accordingly, modeling these networks allows us to study a well engineered system at the component level—a level of modeling not yet possible for more complex mammalian systems, for which detailed structural information is scarce.

## Acknowledgments

This work relies on information supplied by the hard work of many experimentalists. We would especially like to acknowledge the effort and dedication of Peter Getting who devoted 12 years to understanding the organization of the Tritonia network of 14 neurons. We also thank Hewlett-Packard for computing support, and DARPA and MOSIS for chip fabrication. This work was sponsored by the Office of Naval Research, the System Development Foundation, and the NSF (EET-8700064 to J.B.).

## References

Eisen, Judith S. and Marder, Eve (1982). Mechanisms underlying pattern generation in lobster stomatogastric ganglion as determined by selective inactivation of identified neurons. III. Synaptic connections of electrically coupled pyloric neurons. *J. Neurophysiol.* 48:1392–1415.

Getting, Peter A. and Dekin, Michael S. (1985). *Tritonia* swimming: A model system for integration within rhythmic motor systems. In Allen I. Selverston (Ed.), *Model Neural Networks and Behavior*, New York, NY: Plenum Press.

Mead, Carver A. (in press). *Analog VLSI and Neural Systems.* Reading, MA: Addison-Wesley.

Miller, John P. and Selverston, Allen I. (1985). Neural Mechanisms for the production of the lobster pyloric motor pattern. In Allen I. Selverston (Ed.), *Model Neural Networks and Behavior*, New York, NY: Plenum Press.

Miller, R. F. and Dacheux, R. F. (1976). Synaptic organization and ionic basis of on and off channels in mudpuppy retina. *J. Gen. Physiol.* 67:639–690.

Robertson, R. M. and Pearson, K. G. (1985). Neural circuits in the flight system of the locust. *J. Neurophysiol.* 53:110–128.

Selverston, Allen I. and Moulins, Maurice (1985). Oscillatory neural networks. *Ann. Rev. Physiol.* 47:29–48.

Wilson, M. and Bower, J. M. (in press). Simulation of large scale neuronal networks. In C. Koch and I. Segev (Eds.), *Methods in Neuronal Modeling: From Synapses to Networks*, Cambridge, MA: MIT Press.
